# A lattice filter model of the visual pathway

**Karol Gregor**        **Dmitri B. Chklovskii**
Janelia Farm Research Campus, HHMI
19700 Helix Drive, Ashburn, VA
{gregork, mitya}@janelia.hhmi.org

## Abstract

Early stages of visual processing are thought to decorrelate, or whiten, the incoming temporally varying signals. Motivated by the cascade structure of the visual pathway (retina $\to$ lateral geniculate nucelus (LGN) $\to$ primary visual cortex, V1) we propose to model its function using lattice filters - signal processing devices for stage-wise decorrelation of temporal signals. Lattice filter models predict neuronal responses consistent with physiological recordings in cats and primates. In particular, they predict temporal receptive fields of two different types resembling so-called lagged and non-lagged cells in the LGN. Moreover, connection weights in the lattice filter can be learned using Hebbian rules in a stage-wise sequential manner reminiscent of the neuro-developmental sequence in mammals. In addition, lattice filters can model visual processing in insects. Therefore, lattice filter is a useful abstraction that captures temporal aspects of visual processing.

Our sensory organs face an ongoing barrage of stimuli from the world and must transmit as much information about them as possible to the rest of the brain [1]. This is a formidable task because, in sensory modalities such as vision, the dynamic range of natural stimuli (more than three orders of magnitude) greatly exceeds the dynamic range of relay neurons (less than two orders of magnitude) [2]. The reason why high fidelity transmission is possible at all is that the continuity of objects in the physical world leads to correlations in natural stimuli, which imply redundancy. In turn, such redundancy can be eliminated by compression performed by the front end of the visual system leading to the reduction of the dynamic range [3, 4].

A compression strategy appropriate for redundant natural stimuli is called predictive coding [5, 6, 7]. In predictive coding, a prediction of the incoming signal value is computed from past values delayed in the circuit. This prediction is subtracted from the actual signal value and only the prediction error is transmitted. In the absence of transmission noise such compression is lossless as the original signal could be decoded on the receiving end by inverting the encoder. If predictions are accurate, the dynamic range of the error is much smaller than that of the natural stimuli. Therefore, minimizing dynamic range using predictive coding reduces to optimizing prediction.

Experimental support for viewing the front end of the visual system as a predictive encoder comes from the measurements of receptive fields [6, 7]. In particular, predictive coding suggests that, for natural stimuli, the temporal receptive fields should be biphasic and the spatial receptive fields - center-surround. These predictions are born out by experimental measurements in retinal ganglion cells, [8], lateral geniculate nucleus (LGN) neurons [9] and fly second order visual neurons called large monopolar cells (LMCs) [2]. In addition, the experimentally measured receptive fields vary with signal-to-noise ratio as would be expected from optimal prediction theory [6]. Furthermore, experimentally observed whitening of the transmitted signal [10] is consistent with removing correlated components from the incoming signals [11].

As natural stimuli contain correlations on time scales greater than hundred milliseconds, experimentally measured receptive fields of LGN neurons are equally long [12]. Decorrelation over such long time scales requires equally long delays. How can such extended receptive field be produced by

biological neurons and synapses whose time constants are typically less than hundred milliseconds [13]?

The field of signal processing offers a solution to this problem in the form of a device called a lattice filter, which decorrelates signals in stages, sequentially adding longer and longer delays [14, 15, 16, 17]. Motivated by the cascade structure of visual systems [18], we propose to model decorrelation in them by lattice filters. Naturally, visual systems are more complex than lattice filters and perform many other operations. However, we show that the lattice filter model explains several existing observations in vertebrate and invertebrate visual systems and makes testable predictions. Therefore, we believe that lattice filters provide a convenient abstraction for modeling temporal aspects of visual processing.

This paper is organized as follows. First, we briefly summarize relevant results from linear prediction theory. Second, we explain the operation of the lattice filter in discrete and continuous time. Third, we compare lattice filter predictions with physiological measurements.

# 1   Linear prediction theory

Despite the non-linear nature of neurons and synapses, the operation of some neural circuits in vertebrates [19] and invertebrates [20] can be described by a linear systems theory. The advantage of linear systems is that optimal circuit parameters may be obtained analytically and the results are often intuitively clear. Perhaps not surprisingly, the field of signal processing relies heavily on the linear prediction theory, offering a convenient framework [15, 16, 17]. Below, we summarize the results from linear prediction that will be used to explain the operation of the lattice filter.

Consider a scalar sequence $y = \{y_t\}$ where time $t = 1, \ldots, n$. Suppose that $y_t$ at each time point depends on side information provided by vector $\mathbf{z_t}$. Our goal is to generate a series of linear predictions, $\hat{y}_t$ from the vector $\mathbf{z_t}$, $\hat{y}_t = \mathbf{w} \cdot \mathbf{z}_t$. We define a prediction error as:

$$e_t = y_t - \hat{y}_t = y_t - \mathbf{w} \cdot \mathbf{z}_t \tag{1}$$

and look for values of $\mathbf{w}$ that minimize mean squared error:

$$\langle e^2 \rangle = \frac{1}{n_t} \sum_t e_t^2 = \frac{1}{n_t} \sum_t (y_t - \mathbf{w} \cdot \mathbf{z}_t)^2. \tag{2}$$

The weight vector, $\mathbf{w}$ is optimal for prediction of sequence $y$ from sequence $\mathbf{z}$ if and only if the prediction error sequence $e = y - \mathbf{w} \cdot \mathbf{z}$ is orthogonal to each component of vector $\mathbf{z}$:

$$\langle e\mathbf{z} \rangle = \mathbf{0}. \tag{3}$$

When the whole series $y$ is given in advance, i.e. in the offline setting, these so-called normal equations can be solved for $\mathbf{w}$, for example, by Gaussian elimination [21]. However, in signal processing and neuroscience applications, another setting called online is more relevant: At every time step $t$, prediction $\hat{y}_t$ must be made using only current values of $\mathbf{z_t}$ and $\mathbf{w}$. Furthermore, after a prediction is made, $\mathbf{w}$ is updated based on the prediction $\hat{y}_t$ and observed $y_t, \mathbf{z_t}$ .

In the online setting, an algorithm called stochastic gradient descent is often used, where, at each time step, $\mathbf{w}$ is updated in the direction of negative gradient of $e_t^2$:

$$\mathbf{w} \to \mathbf{w} - \eta \nabla_{\mathbf{w}} (y_t - \mathbf{w} \cdot \mathbf{z}_t)^2. \tag{4}$$

This leads to the following weight update, known as least mean square (LMS) [15], for predicting sequence $y$ from sequence $\mathbf{z}$:

$$\mathbf{w} \to \mathbf{w} + \eta e_t \mathbf{z}_t, \tag{5}$$

where $\eta$ is the learning rate. The value of $\eta$ represents the relative influence of more recent observations compared to more distant ones. The larger the learning rate the faster the system adapts to recent observations and less past it remembers.

In this paper, we are interested in predicting a current value $x_t$ of sequence $x$ from its past values $x_{t-1}, \ldots, x_{t-k}$ restricted by the prediction order $k > 0$:

$$\hat{x}_t = \mathbf{w}^k \cdot (x_{t-1}, \ldots, x_{t-k})^T. \tag{6}$$

This problem is a special case of the online linear prediction framework above, where $y_t = x_t$, $\mathbf{z}_t = (x_{t-1}, \ldots, x_{t-k})^T$. Then the gradient update is given by:

$$\mathbf{w} \rightarrow \mathbf{w}^k + \eta e_t (x_{t-1}, \ldots, x_{t-k})^T. \tag{7}$$

While the LMS algorithm can find the weights that optimize linear prediction (6), the filter $\mathbf{w}^k$ has a long temporal extent making it difficult to implement with neurons and synapses.

## 2 Lattice filters

One way to generate long receptive fields in circuits of biological neurons is to use a cascade architecture, known as the lattice filter, which calculates optimal linear predictions for temporal sequences and transmits prediction errors [14, 15, 16, 17]. In this section, we explain the operation of a discrete-time lattice filter, then adapt it to continuous-time operation.

### 2.1 Discrete-time implementation

The first stage of the lattice filter, Figure 1, calculates the error of the first order optimal prediction (i.e. only using the preceding element of the sequence), the second stage uses the output of the first stage and calculates the error of the second order optimal prediction (i.e. using only two previous values) etc. To make such stage-wise error computations possible the lattice filter calculates at every stage not only the error of optimal prediction of $x_t$ from past values $x_{t-1}, \ldots, x_{t-k}$, called forward error,

$$f_t^k = x_t - \mathbf{w}^k \cdot (x_{t-1}, \ldots, x_{t-k})^T, \tag{8}$$

but, perhaps non-intuitively, also the error of optimal prediction of a past value $x_{t-k}$ from the more recent values $x_{t-k+1}, \ldots, x_t$, called backward error:

$$b_t^k = x_{t-k} - \mathbf{w}'^k \cdot (x_{t-k+1}, \ldots, x_t)^T, \tag{9}$$

where $\mathbf{w}^k$ and $\mathbf{w}'^k$ are the weights of the optimal prediction.

For example, the first stage of the filter calculates the forward error $f_t^1$ of optimal prediction of $x_t$ from $x_{t-1}$: $f_t^1 = x_t - u^1 x_{t-1}$ as well as the backward error $b_t^1$ of optimal prediction of $x_{t-1}$ from $x_t$: $b_t^1 = x_{t-1} - v^1 x_t$, Figure 1. Here, we assume that coefficients $u^1$ and $v^1$ that give optimal linear prediction are known and return to learning them below.

Each following stage of the lattice filter performs a stereotypic operation on its inputs, Figure 1. The $k$-th stage $(k > 1)$ receives forward, $f_t^{k-1}$, and backward, $b_t^{k-1}$, errors from the previous stage, delays backward error by one time step and computes a forward error:

$$f_t^k = f_t^{k-1} - u^k b_{t-1}^{k-1} \tag{10}$$

of the optimal linear prediction of $f_t^{k-1}$ from $b_{t-1}^{k-1}$. In addition, each stage computes a backward error

$$b_t^k = b_{t-1}^{k-1} - v^k f_t^{k-1} \tag{11}$$

of the optimal linear prediction of $b_{t-1}^{k-1}$ from $f_t^{k-1}$.

As can be seen in Figure 1, the lattice filter contains forward prediction error (top) and backward prediction error (bottom) branches, which interact at every stage via cross-links. Operation of the lattice filter can be characterized by the linear filters acting on the input, $x$, to compute forward or backward errors of consecutive order, so called prediction-error filters (blue bars in Figure 1). Because of delays in the backward error branch the temporal extent of the filters grows from stage to stage.

In the next section, we will argue that prediction-error filters correspond to the measurements of temporal receptive fields in neurons. For detailed comparison with physiological measurements we will use the result that, for bi-phasic prediction-error filters, such as the ones in Figure 1, the first bar of the forward prediction-error filter has larger weight, by absolute value, than the combined weights of the remaining coefficients of the corresponding filter. Similarly, in backward prediction-error filters, the last bar has greater weight than the rest of them combined. This fact arises from the observation that forward prediction-error filters are minimum phase, while backward prediction-error filters are maximum phase [16, 17].

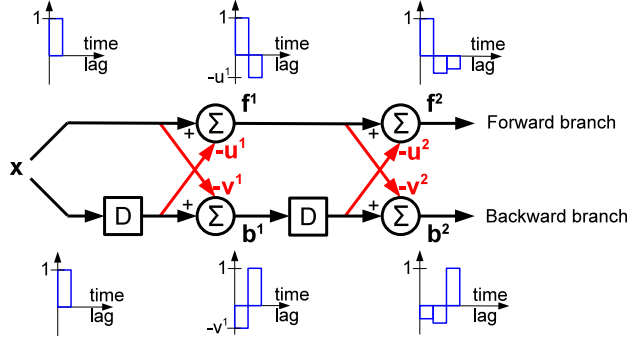

Figure 1: **Discrete-time lattice filter performs stage-wise computation of forward and backward prediction errors.** In the first stage, the optimal prediction of $x_t$ from $x_{t-1}$ is computed by delaying the input by one time step and multiplying it by $u^1$. The upper summation unit subtracts the predicted $x_t$ from the actual value and outputs prediction error $f_t^1$. Similarly, the optimal prediction of $x_{t-1}$ from $x_t$ is computed by multiplying the input by $v^1$. The lower summation unit subtracts the optimal prediction from the actual value and outputs backward error $b_t^1$. In each following stage $k$, the optimal prediction of $f_t^{k-1}$ from $b_t^{k-1}$ is computed by delaying $b_t^{k-1}$ by one time step and multiplying it by $u^k$. The upper summation unit subtracts the prediction from the actual $f_t^{k-1}$ and outputs prediction error $f_t^k$. Similarly, the optimal prediction of $b_{t-1}^{k-1}$ from $f_t^{k-1}$ is computed by multiplying it by $u^k$. The lower summation unit subtracts the optimal prediction from the actual value and outputs backward error $b_t^k$. Black connections have unitary weights and red connections have learnable negative weights. One can view forward and backward error calculations as applications of so-called prediction-error filters (blue) to the input sequence. Note that the temporal extent of the filters gets longer from stage to stage.

Next, we derive a learning rule for finding optimal coefficients $u$ and $v$ in the online setting. The $u^k$ is used for predicting $f_t^{k-1}$ from $b_{t-1}^{k-1}$ to obtain error $f_t^k$. By substituting $y_t = f_t^{k-1}$, $\mathbf{z}_t = b_{t-1}^{k-1}$ and $e_t = f_t^k$ into (5) the update of $u^k$ becomes

$$u^k \to u^k + \eta f_t^k b_{t-1}^{k-1}. \tag{12}$$

Similarly, $v^k$ is updated by

$$v^k \to v^k + \eta b_t^k f_t^{k-1}. \tag{13}$$

Interestingly, the updates of the weights are given by the product of the activities of outgoing and incoming nodes of the corresponding cross-links. Such updates are known as Hebbian learning rules thought to be used by biological neurons [22, 23].

Finally, we give a simple proof that, in the offline setting when the entire sequence $x$ is known, $f^k$ and $b^k$, given by equations (10, 11), are indeed errors of optimal $k$-th order linear prediction. Let $D$ be one step time delay operator $(Dx)_t = x_{t-1}$. The induction statement at $k$ is that $f^k$ and $b^k$ are $k$-th order forward and backward errors of optimal linear prediction which is equivalent to $f^k$ and $b^k$ being of the form $f^k = x - w_1^k Dx - \ldots - w_k^k D^k x$ and $b^k = D^k x - w_1'^k D^{k-1} x - \ldots - w_k'^k x$ and, from normal equations (3), satisfying $\langle f^k D^i x \rangle = 0$ and $\langle Db^k D^i x \rangle = \langle b^k D^{i-1} x \rangle = 0$ for $i = 1, \ldots, k$. That this is true for $k = 1$ directly follows from the definition of $f^1$ and $b^1$. Now we assume that this is true for $k - 1 \geq 1$ and show it is true for $k$. It is easy to see from the forms of $f^{k-1}$ and $b^{k-1}$ and from $f^k = f^{k-1} - u^k Db^{k-1}$ that $f^k$ has the correct form $f^k = x - w_1^k Dx - \ldots - w_k^k D^k x$. Regarding orthogonality for $i = 1, \ldots, k-1$ we have $\langle f^k D^i x \rangle = \langle (f^{k-1} - u^k Db^{k-1}) D^i x \rangle = \langle f^{k-1} D^i x \rangle - u^k \langle (Db^{k-1}) D^i x \rangle = 0$ using the induction assumptions of orhogonality at $k - 1$. For the remaining $i = k$ we note that $f^k$ is the error of the optimal linear prediction of $f^{k-1}$ from $Db^{k-1}$ and therefore $0 = \langle f^k Db^{k-1} \rangle = \langle f^k (D^k x - w_1'^{k-1} D^{k-1} x - \ldots + w_{k-1}'^{k-1} Dx) \rangle = \langle f^k D^k x \rangle$ as desired. The $b^k$ case can be proven similarly.

## 2.2 Continuous-time implementation

The last hurdle remaining for modeling neuronal circuits which operate in continuous time with a lattice filter is its discrete-time operation. To obtain a continuous-time implementation of the lattice

filter we cannot simply take the time step size to zero as prediction-error filters would become infinitesimally short. Here, we adapt the discrete-time lattice filter to continous-time operation in two steps.

First, we introduce a discrete-time Laguerre lattice filter [24, 17] which uses Laguerre polynomials rather than the shift operator to generate its basis functions, Figure 2. The input signal passes through a leaky integrator whose leakage constant $\alpha$ defines a time-scale distinct from the time step (14). A delay, $D$, at every stage is replaced by an all-pass filter, $L$, (15) with the same constant $\alpha$, which preserves the magnitude of every Fourier component of the input but shifts its phase in a frequency dependent manner. Such all-pass filter reduces to a single time-step delay when $\alpha = 0$. The optimality of a general discrete-time Laguerre lattice filter can be proven similarly to that for the discrete-time filter, simply by replacing operator $D$ with $L$ in the proof of section 2.1.

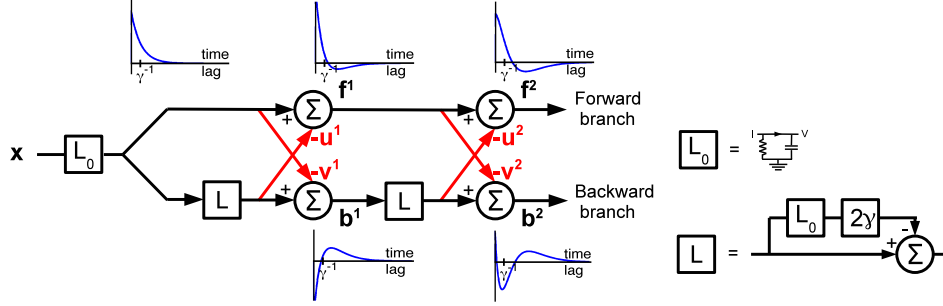

Figure 2: **Continuous-time lattice filter using Laguerre polynomials.** Compared to the discrete-time version, it contains a leaky integrator, $L_0$,(16) and replaces delays with all-pass filters, $L$, (17).

Second, we obtain a continuous-time formulation of the lattice filter by replacing $t - 1 \rightarrow t - \delta t$, defining the inverse time scale $\gamma = (1 - \alpha)/\delta t$ and taking the limit $\delta t \rightarrow 0$ while keeping $\gamma$ fixed. As a result $L_0$ and $L$ are given by:

$$
\begin{array}{llr}
\text{Discrete time} & \text{Continuous time} & \\
L_0(x)_t = \alpha L_0(x)_{t-1} + x_t \quad (14) & dL_0(x)/dt = -\gamma L_0(x) + x & (16) \\
L(x)_t = \alpha(L(x)_{t-1} - x_t) + x_{t-1} \quad (15) & L(x) = x - 2\gamma L_0(x) & (17)
\end{array}
$$

Representative impulse responses of the continuous Laguerre filter are shown in Figure 2. Note that, similarly to the discrete-time case, the area under the first (peak) phase is greater than the area under the second (rebound) phase in the forward branch and the opposite is true in the backward branch. Moreover, the temporal extent of the rebound is greater than that of the peak not just in the forward branch like in the basic discrete-time implementation but also in the backward branch. As will be seen in the next section, these predictions are confirmed by physiological recordings.

# 3 Experimental evidence for the lattice filter in visual pathways

In this section we demonstrate that physiological measurements from visual pathways in vertebrates and invertebrates are consistent with the predictions of the lattice filter model. For the purpose of modeling visual pathways, we identify summation units of the lattice filter with neurons and propose that neural activity represents forward and backward errors. In the fly visual pathway neuronal activity is represented by continuously varying graded potentials. In the vertebrate visual system, all neurons starting with ganglion cells are spiking and we identify their firing rate with the activity in the lattice filter.

## 3.1 Mammalian visual pathway

In mammals, visual processing is performed in stages. In the retina, photoreceptors synapse onto bipolar cells, which in turn synapse onto retinal ganglion cells (RGCs). RGCs send axons to the LGN, where they synapse onto LGN relay neurons projecting to the primary visual cortex, V1. In addition to this feedforward pathway, at each stage there are local circuits involving (usually inhibitory) inter-neurons such as horizontal and amacrine cells in the retina. Neurons of each class

come in many types, which differ in their connectivity, morphology and physiological response. The bewildering complexity of these circuits has posed a major challenge to visual neuroscience.

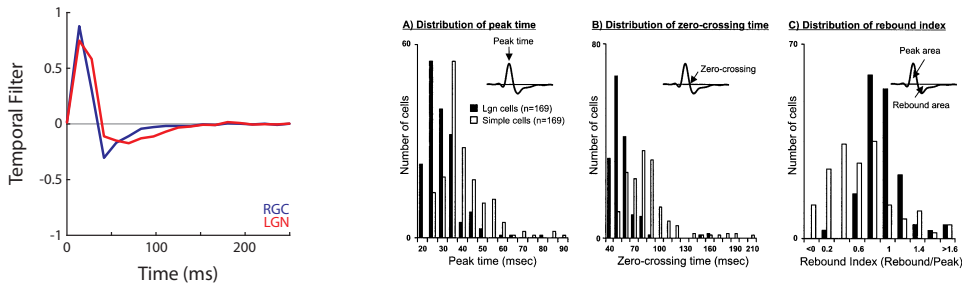

Figure 3: **Electrophysiologically measured temporal receptive fields get progressively longer along the cat visual pathway.** Left: A cat LGN cell (red) has a longer receptive field than a corresponding RGC cell (blue) (adapted from [12] which also reports population data). Right (A,B): Extent of the temporal receptive fields of simple cells in cat V1 is greater than that of corresponding LGN cells as quantified by the peak (A) and zero-crossing (B) times. Right (C): In the temporal receptive fields of cat LGN and V1 cells the peak can be stronger or weaker than the rebound (adapted from [25]).

Here, we point out several experimental observations related to temporal processing in the visual system consistent with the lattice filter model. First, measurements of temporal receptive fields demonstrate that they get progressively longer at each consecutive stage: i) LGN neurons have longer receptive fields than corresponding pre-synaptic ganglion cells [12], Figure 3left; ii) simple cells in V1 have longer receptive fields than corresponding pre-synaptic LGN neurons [25], Figure 3rightA,B. These observation are consistent with the progressively greater temporal extent of the prediction-error filters (blue plots in Figure 2).

Second, the weight of the peak (integrated area under the curve) may be either greater or less than that of the rebound both in LGN relay cells [26] and simple cells of V1 [25], Figure 3right(C). Neurons with peak weight exceeding that of rebound are often referred to as non-lagged while the others are known as lagged found both in cat [27, 28, 29] and monkey [30]. The reason for this becomes clear from the response to a step stimulus, Figure 4(top).

By comparing experimentally measured receptive fields with those of the continuous lattice filter, Figure 4, we identify non-lagged neurons with the forward branch and lagged neurons with the backward branch. Another way to characterize step-stimulus response is whether the sign of the transient is the same (non-lagged) or different (lagged) relative to sustained response.

Third, measurements of cross-correlation between RGCs and LGN cell spikes in lagged and non-lagged neurons reveals a difference of the transfer function indicative of the difference in underlying circuitry [31]. This is consistent with the backward branch circuit of the Laguerre lattice filter, Figure 2, being different then that of the forward branch (which results in different transfer function). In particular, a combination of different glutamate receptors such as AMPA and NMDA, as well as GABA receptors are thought to be responsible for observed responses in lagged cells [32]. However, further investigation of the corresponding circuitry, perhaps using connectomics technology, is desirable.

Fourth, the cross-link weights of the lattice filter can be learned using Hebbian rules, (12,13) which are biologically plausible [22, 23]. Interestingly, if these weights are learned sequentially, starting from the first stage, they do not need to be re-learned when additional stages are added or learned. This property maps naturally on the fact that in the course of mammalian development the visual pathway matures in a stage-wise fashion - starting with the retina, then LGN, then V1 - and implying that the more peripheral structures do not need to adapt to the maturation of the downstream ones.

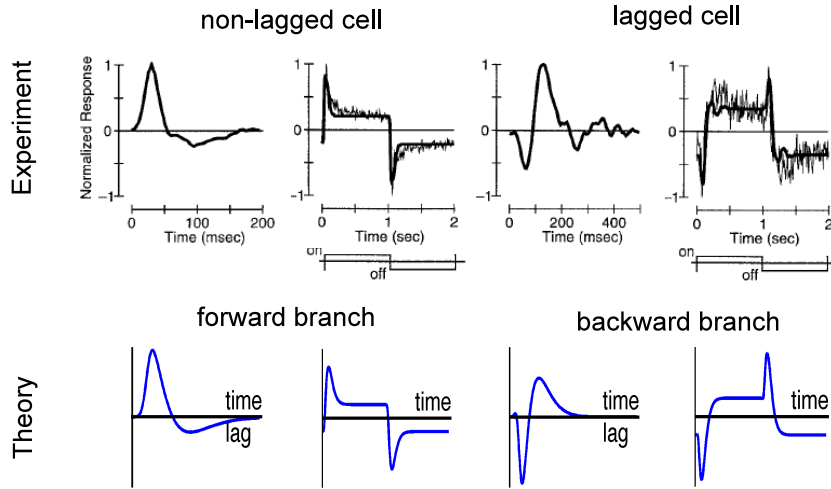

Figure 4: **Comparison of electrophysiologically measured responses of cat LGN cells with the continuous-time lattice filter model.** Top: Experimentally measured temporal receptive fields and step-stimulus responses of LGN cells (adapted from [26]). Bottom: Typical examples of responses in the continuous-time lattice filter model. Lattice filter coefficients were $u^1 = v^1 = 0.4, u^2 = v^2 = 0.2$ and $1/\gamma = 50$ms to model the non-lagged cell and $u^1 = v^1 = u^2 = v^2 = 0.2$ and $1/\gamma = 60$ms to model the lagged cell. To model photoreceptor contribution to the responses, an additional leaky integrator $L_0$ was added to the circuit of Figure 2.

While Hebbian rules are biologically plausible, one may get an impression from Figure 2 that they must apply to inhibitory cross-links. We point out that this circuit is meant to represent only the computation performed rather than the specific implementation in terms of neurons. As the same linear computation can be performed by circuits with a different arrangement of the same components, there are multiple implementations of the lattice filter. For example, activity of non-lagged OFF cells may be seen as representing minus forward error. Then the cross-links between the non-lagged OFF pathway and the lagged ON pathway would be excitatory. In general, classification of cells into lagged and non-lagged seems independent of their ON/OFF and X/Y classification [31, 28, 29], but see[33].

## 3.2   Insect visual pathway

In insects, two cell types, L1 and L2, both post-synaptic to photoreceptors play an important role in visual processing. Physiological responses of L1 and L2 indicate that they decorrelate visual signals by subtracting their predictable parts. In fact, receptive fields of these neurons were used as the first examples of predictive coding in neuroscience [6]. Yet, as the numbers of synapses from photoreceptors to L1 and L2 are the same [34] and their physiological properties are similar, it has been a mystery why insects, have not just one but a pair of such seemingly redundant neurons per facet. Previously, it was suggested that L1 and L2 provide inputs to the two pathways that map onto ON and OFF pathways in the vertebrate retina [35, 36].

Here, we put forward a hypothesis that the role of L1 and L2 in visual processing is similar to that of the two branches of the lattice filter. We do not incorporate the ON/OFF distinction in the effectively linear lattice filter model but anticipate that such combined description will materialize in the future. As was argued in Section 2, in forward prediction-error filters, the peak has greater weight than the rebound, while in backward prediction-error filters the opposite is true. Such difference implies that in response to a step-stimulus the signs of sustained responses compared to initial transients are different between the branches. Indeed, $Ca^{2+}$ imaging shows that responses of L1 and L2 to step-stimulus are different as predicted by the lattice filter model [35], Figure 5b. Interestingly, the activity of L1 seems to represent minus forward error and L2 - plus backward error, suggesting that the lattice filter cross-links are excitatory. To summarize, the predictions of the lattice filter model seem to be consistent with the physiological measurements in the fly visual system and may help understand its operation.

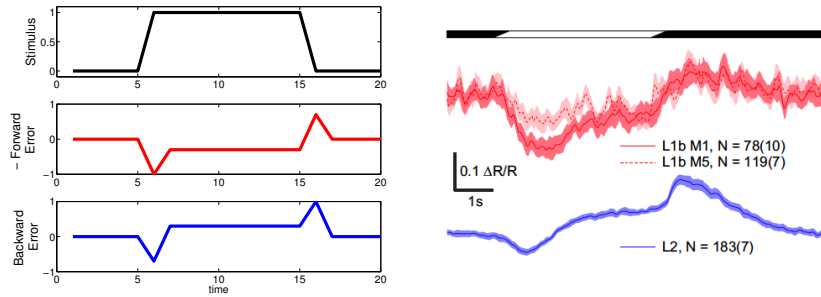

Figure 5: **Response of the lattice filter and fruit fly LMCs to a step-stimulus.** Left: Responses of the first order discrete-time lattice filter to a step stimulus. Right: Responses of fly L1 and L2 cells to a moving step stimulus (adapted from [35]). Predicted and the experimentally measured responses have qualitatively the same shape: a transient followed by sustained response, which has the same sign for the forward error and L1 and the opposite sign for the backward error and L2.

## 4    Discussion

Motivated by the cascade structure of the visual pathway, we propose to model its operation with the lattice filter. We demonstrate that the predictions of the continuous-time lattice filter model are consistent with the course of neural development and the physiological measurement in the LGN, V1 of cat and monkey, as well as fly LMC neurons. Therefore, lattice filters may offer a useful abstraction for understanding aspects of temporal processing in visual systems of vertebrates and invertebrates.

Previously, [11] proposed that lagged and non-lagged cells could be a result of rectification by spiking neurons. Although we agree with [11] that LGN performs temporal decorrelation, our explanation does not rely on non-linear processing but rather on the cascade architecture and, hence, is fundamentally different. Our model generates the following predictions that are not obvious in [11]: i) Not only are LGN receptive fields longer than RGC but also V1 receptive fields are longer than LGN; ii) Even a linear model can generate a difference in the peak/rebound ratio; iii) The circuit from RGC to LGN should be different for lagged and non-lagged cells consistent with [31]; iv) The lattice filter circuit can self-organize using Hebbian rules, which gives a mechanistic explanation of receptive fields beyond the normative framework of [11].

In light of the redundancy reduction arguments given in the introduction, we note that, if the only goal of the system were to compress incoming signals using a given number of lattice filter stages, then after the compression is peformed only one kind of prediction errors, forward or backward needs to be transmitted. Therefore, having two channels, in the absence of noise, may seem redundant. However, transmitting both forward and backward errors gives one the flexibility to continue decorrelation further by adding stages performing relatively simple operations.

We are grateful to D.A. Butts, E. Callaway, M. Carandini, D.A. Clark, J.A. Hirsch, T. Hu, S.B. Laughlin, D.N. Mastronarde, R.C. Reid, H. Rouault, A. Saul, L. Scheffer, F.T. Sommer, X. Wang for helpful discussions.

## References

[1] F. Rieke, D. Warland, R.R. van Steveninck, and W. Bialek. *Spikes: exploring the neural code*. MIT press, 1999.

[2] S.B. Laughlin. Matching coding, circuits, cells, and molecules to signals: general principles of retinal design in the fly's eye. *Progress in retinal and eye research*, 13(1):165–196, 1994.

[3] F. Attneave. Some informational aspects of visual perception. *Psychological review*, 61(3):183, 1954.

[4] H. Barlow. Redundancy reduction revisited. *Network: Comp in Neural Systems*, 12(3):241–253, 2001.

[5] R.M. Gray. *Linear Predictive Coding and the Internet Protocol*. Now Publishers, 2010.

[6] MV Srinivasan, SB Laughlin, and A. Dubs. Predictive coding: a fresh view of inhibition in the retina. *Proceedings of the Royal Society of London. Series B. Biological Sciences*, 216(1205):427–459, 1982.

[7] T. Hosoya, S.A. Baccus, and M. Meister. Dynamic predictive coding by the retina. *Nature*, 436:71, 2005.

[8] HK Hartline, H.G. Wagner, and EF MacNichol Jr. The peripheral origin of nervous activity in the visual system. *Studies on excitation and inhibition in the retina: a collection of papers from the laboratories of H. Keffer Hartline*, page 99, 1974.

[9] N.A. Lesica, J. Jin, C. Weng, C.I. Yeh, D.A. Butts, G.B. Stanley, and J.M. Alonso. Adaptation to stimulus contrast and correlations during natural visual stimulation. *Neuron*, 55(3):479–491, 2007.

[10] Y. Dan, J.J. Atick, and R.C. Reid. Efficient coding of natural scenes in the lateral geniculate nucleus: experimental test of a computational theory. *The Journal of Neuroscience*, 16(10):3351–3362, 1996.

[11] D.W. Dong and J.J. Atick. Statistics of natural time-varying images. *Network: Computation in Neural Systems*, 6(3):345–358, 1995.

[12] X. Wang, J.A. Hirsch, and F.T. Sommer. Recoding of sensory information across the retinothalamic synapse. *The Journal of Neuroscience*, 30(41):13567–13577, 2010.

[13] C. Koch. *Biophysics of computation: information processing in single neurons*. Oxford Univ Press, 2005.

[14] F. Itakura and S. Saito. On the optimum quantization of feature parameters in the parcor speech synthesizer. In *Conference Record, 1972 International Conference on Speech Communication and Processing, Boston, MA*, pages 434–437, 1972.

[15] B. Widrow and S.D. Stearns. *Adaptive signal processing*. Prentice-Hall, Inc. Englewood Cliffs, NJ, 1985.

[16] S. Haykin. *Adaptive filter theory*. Prentice-Hall, Englewood-Cliffs, NJ, 2003.

[17] A.H. Sayed. *Fundamentals of adaptive filtering*. Wiley-IEEE Press, 2003.

[18] D.J. Felleman and D.C. Van Essen. Distributed hierarchical processing in the primate cerebral cortex. *Cerebral cortex*, 1(1):1–47, 1991.

[19] X. Wang, F.T. Sommer, and J.A. Hirsch. Inhibitory circuits for visual processing in thalamus. *Current Opinion in Neurobiology*, 2011.

[20] SB Laughlin, J. Howard, and B. Blakeslee. Synaptic limitations to contrast coding in the retina of the blowfly calliphora. *Proceedings of the Royal society of London. Series B. Biological sciences*, 231(1265):437–467, 1987.

[21] D.C. Lay. *Linear Algebra and Its Applications*. Addison-Wesley/Longman, New York/London, 2000.

[22] D.O. Hebb. *The organization of behavior: A neuropsychological theory*. Lawrence Erlbaum, 2002.

[23] O. Paulsen and T.J. Sejnowski. Natural patterns of activity and long-term synaptic plasticity. *Current opinion in neurobiology*, 10(2):172–180, 2000.

[24] Z. Fejzo and H. Lev-Ari. Adaptive laguerre-lattice filters. *Signal Processing, IEEE Transactions on*, 45(12):3006–3016, 1997.

[25] J.M. Alonso, W.M. Usrey, and R.C. Reid. Rules of connectivity between geniculate cells and simple cells in cat primary visual cortex. *The Journal of Neuroscience*, 21(11):4002–4015, 2001.

[26] D. Cai, G.C. Deangelis, and R.D. Freeman. Spatiotemporal receptive field organization in the lateral geniculate nucleus of cats and kittens. *Journal of Neurophysiology*, 78(2):1045–1061, 1997.

[27] D.N. Mastronarde. Two classes of single-input x-cells in cat lateral geniculate nucleus. i. receptive-field properties and classification of cells. *Journal of Neurophysiology*, 57(2):357–380, 1987.

[28] J. Wolfe and L.A. Palmer. Temporal diversity in the lateral geniculate nucleus of cat. *Visual neuroscience*, 15(04):653–675, 1998.

[29] AB Saul and AL Humphrey. Spatial and temporal response properties of lagged and nonlagged cells in cat lateral geniculate nucleus. *Journal of Neurophysiology*, 64(1):206–224, 1990.

[30] A.B. Saul. Lagged cells in alert monkey lateral geniculate nucleus. *Visual neurosci*, 25:647–659, 2008.

[31] D.N. Mastronarde. Two classes of single-input x-cells in cat lateral geniculate nucleus. ii. retinal inputs and the generation of receptive-field properties. *Journal of Neurophysiology*, 57(2):381–413, 1987.

[32] P. Heggelund and E. Hartveit. Neurotransmitter receptors mediating excitatory input to cells in the cat lateral geniculate nucleus. i. lagged cells. *Journal of neurophysiology*, 63(6):1347–1360, 1990.

[33] J. Jin, Y. Wang, R. Lashgari, H.A. Swadlow, and J.M. Alonso. Faster thalamocortical processing for dark than light visual targets. *The Journal of Neuroscience*, 31(48):17471–17479, 2011.

[34] M. Rivera-Alba, S.N. Vitaladevuni, Y. Mischenko, Z. Lu, S. Takemura, L. Scheffer, I.A. Meinertzhagen, D.B. Chklovskii, and G.G. de Polavieja. Wiring economy and volume exclusion determine neuronal placement in the drosophila brain. *Current Biology*, 21(23):2000–5, 2011.

[35] D.A. Clark, L. Bursztyn, M.A. Horowitz, M.J. Schnitzer, and T.R. Clandinin. Defining the computational structure of the motion detector in drosophila. *Neuron*, 70(6):1165–1177, 2011.

[36] M. Joesch, B. Schnell, S.V. Raghu, D.F. Reiff, and A. Borst. On and off pathways in drosophila motion vision. *Nature*, 468(7321):300–304, 2010.

